# AN ADAPTIVE NETWORK THAT LEARNS SEQUENCES OF TRANSITIONS

C. L. Winter

Science Applications International Corporation
5151 East Broadway, Suite 900
Tucson, Arizona 85711

## ABSTRACT

We describe an adaptive network, $TIN^2$, that learns the transition function of a sequential system from observations of its behavior. It integrates two subnets, TIN-1 (Winter, Ryan and Turner, 1987) and TIN-2. TIN-2 constructs state representations from examples of system behavior, and its dynamics are the main topics of the paper. TIN-1 abstracts transition functions from noisy state representations and environmental data during training, while in operation it produces sequences of transitions in response to variations in input. Dynamics of both nets are based on the Adaptive Resonance Theory of Carpenter and Grossberg (1987). We give results from an experiment in which $TIN^2$ learned the behavior of a system that recognizes strings with an even number of 1's .

## INTRODUCTION

Sequential systems respond to variations in their input environment with sequences of activities. They can be described in two ways. A black box description characterizes a system as an input-output function, $m = B(\underline{u})$, mapping a string of input symbols, $\underline{u}$, into a single output symbol, m. A sequential automaton description characterizes a system as a sextuple $(U, M, S, s_0, f, g)$ where U and M are alphabets of input and output symbols, S is a set of states, $s_0$ is an initial state and f and g are transition and output functions respectively. The transition function specifies the current state, $s_t$, as a function of the last state and the current input, $u_t$,

$$s_t = f( s_{t-1}, u_t ) . \qquad (1)$$

In this paper we do not discuss output functions because they are relatively simple. To further simplify discussion, we restrict ourselves to binary input alphabets, although the neural net we describe here can easily be extended to accomodate more complex alphabets.

A common engineering problem is to identify and then simulate the functionality of a system from observations of its behavior. Simulation is straightforward when we can actually observe the internal states of a system, since then the function f can be specified by learning simple associations among internal states and external inputs. In robotic systems, for instance, internal states can often be characterized by such parameters as stepper motor settings, strain gauge values, etc., and so are directly accessible. Artificial neural systems have been found useful in such simulations because they can associate large, possibly noisy state space and input variables with state and output variables (Tolat and Widrow, 1988; Winter, Ryan and Turner, 1987).

Unfortunately, in many interesting cases we must base simulations on a limited set of examples of a system's black box behavior because its internal workings are unobservable. The black box description is not, by itself, much use as a simulation tool since usually it cannot be specified without resorting to infinitely large input-output tables. As an alternative we can try to develop a sequential automaton description of the system by observing regularities in its black box behavior. Artificial neural systems can contribute to the development of physical machines dedicated to system identification because i) frequently state representations must be derived from many noisy input variables, ii) data must usually be processed in continuous time and iii) the explicit dynamics of artificial neural systems can be used as a framework for hardware implementations.

In this paper we give a brief overview of a neural net, $TIN^2$, which learns and processes state transitions from observations of correct black box behavior when the set of observations is large enough to characterize the black box as an automaton. The $TIN^2$ net is based on two component networks. Each uses a modified adaptive resonance circuit (Carpenter and Grossberg, 1987) to associate heterogeneous input patterns. TIN-1 (Winter, Ryan and Turner, 1987) learns and executes transitions when given state representations. It has been used by itself to simulate systems for which explicit state representations are available (Winter, 1988a). TIN-2 is a highly parallel, continuous time implementation of an approach to state representation first outlined by Nerode (1958).

Nerode's approach to system simulation relies upon the fact that every string, $\underline{u}$, moves a machine into a particular state, $s(\underline{u})$, once it has been processed. The $s(\underline{u})$ state can be characterized by putting the system initially into $s(\underline{u})$ (by processing $\underline{u}$) and then presenting a set of experimental strings, $\{\underline{w}_1, ..., \underline{w}_n\}$, for further processing. Experiments consist of observing the output $m_i = B(\underline{u} \cdot \underline{w}_i)$ where $\cdot$ indicates concatenation. A state can then be represented by the entries in a row of a state characterization table, C (Table 1). The rows of C are indexed by strings, $\underline{u}$, its columns are indexed by experiments, $\underline{w}_i$, and its entries are $m_i$. In Table 1 annotations in parentheses indicate nodes (artificial neurons) and subnetworks of TIN-2 equivalent to the corresponding C table entry. During experimentation C expands as    states are

distinguished from one another.  The orchestration of experiments, their selection, the

TABLE 1.  C Table Constructed by TIN-2

|  | $\lambda$ | 0 (Assembly 1) | 1 (Assembly 2) |
|---|---|---|---|
| $\lambda$ | 1 (Node 7) | 0 (Node 2) | 0 (Node 5) |
| 1 | 0 (Node 6) | 0 (Node 9) | 1 (Node 1) |
| 0 | 0 (Node 1) | 1 (Node 6) | 0 (Node 4) |
| 10 | 0 (Node 3) | 0 (Node 2) | 0 (Node 0) |

role of teachers and of the environment have been investigated by Arbib and Zeiger (1969), Arbib and Manes (1974), Gold (1972 and 1978) and Angluin (1987) to name a few.  TIN-2 provides an architecture in which C can be embedded and expanded as necessary.  Collections of nodes within TIN-2 learn to associate triples $(m_i, \underline{u}, \underline{w}_i)$ so that inputting $\underline{u}$ later results in the output of the representation $(m_1, ..., m_n)_{\underline{u}}$ of the state associated with $\underline{u}$.

# TIN-2

TIN-2 is composed of separate assemblies of nodes whose dynamics are such that each assembly comes to correspond to a column in the state characterization table C.  Thus we call them column-assemblies.  Competition among column-assemblies guarantees that nodes of only one assembly, say the $i^{th}$, learn to respond to experimental pattern $\underline{w}_i$.  Hence column-assemblies can be labelled $\underline{w}_1, \underline{w}_2$ and so on, but since labelings are not assigned ahead of time, arbitrarily large sets of experiments can be learned.

The theory of adaptive resonance is implemented in TIN-2 column-assemblies through partitioned adaptive resonance circuits (cf. Ryan, Winter and Turner, 1987).  Adaptive resonance circuits (Grossberg and Carpenter, 1987; Ryan and Winter, 1987) are composed of four collections of nodes: Input, Comparison ($F_1$), Recognition ($F_2$) and Reset.  In TIN-2 Input, Comparison and Reset are split into disjoint m, $\underline{u}$ and $\underline{w}$ partitions.  The net runs in either training or operational mode, and can move from one to the other as required.  The training dynamics of the circuit are such that an $F_2$ node is stimulated by the overall triple $(m, \underline{u}, \underline{w})$, but can be inhibited by a mismatch with any component.  During operation input of $\underline{u}$ recalls the state representation $s(\underline{u}) = (m_1, ..., m_n)_{\underline{u}}$.

Node activity for the $k^{th}$ $F_1$ partition, $F_{1,k}$, $k = m, u, w$, is governed by

$$\tau \, dx_{i,k}/dt = -x_{i,k} + \sum_{j \in F_2} T_{ji} f(y_j) + I_{i,k} \, . \tag{2}$$

Here $\tau < 1$ scales time, $I_{i,k}$ is the value of the $i^{th}$ input node of partition k, $x_{i,k}$ is

activity in the corresponding node of $F_1$ and f is a sigmoid function with range [0, 1]. The elements of $\underline{I}$ are either 1, -1 or 0. The dynamics of the TIN-2 circuit are such that 0 indicates the absence of a symbol, while 1 and -1 represent elements of a binary alphabet. The adaptive feedback filter, T, is a matrix $(T_{ji})$ whose elements, after training, are also 1, -1 or 0.

Activity, $y_j$, in the $j^{th}$ $F_2$ node is driven by

$$\tau \; dy_j/dt = -y_j + [f(y_j) + \Sigma_{u \in F1,u} B_{uj} \, h(x_u) + \Sigma_{w \in F1,w} \, B_w \, h(x_w)$$

$$+ \Sigma_{m \in F1,m} B_{mj} \, h(x_m)] - 4[\Sigma_{\eta \neq j} f(y_\eta) + R_{u,j} + R_w] \; . \tag{3}$$

The feedforward filter B is composed of matrices $(B_{u,j})$, $(B_{m,j})$ and $(B_w)$ whose elements are normalized to the size of the patterns memorized. Note that $(B_w)$ is the same for every node in a given column-assembly, i.e. the rows of $(B_w)$ are all the same. Hence all nodes within a column-assembly learn to respond to the same experimental pattern, $\underline{w}$, and it is in this sense that an assembly evolves to become equivalent to a column in table C. During training the sum $\Sigma_{\eta \neq j} f(y_\eta)$ in (3) runs through the recognition nodes of all TIN-2 column-assemblies. Thus, during training only one $F_2$ node, say the $J^{th}$, can be active at a time across all assemblies. In operation, on the other hand, we remove inhibition due to nodes in other assemblies so that at any time one node in each column-assembly can be active, and an entire state representation can be recalled.

The Reset terms $R_{u,j}$ and $R_w$ in (3) actively inhibit nodes of $F_2$ when mismatches between memory and input occur. $R_{u,j}$ is specific to the $j^{th}$ $F_2$ node,

$$dR_{u,j}/dt = -R_{u,j} + f(y_j) \, f(v \, \| \, \underline{I}_u \, \| - \| \, \underline{P}_{1,u} \, \|) \; . \tag{4}$$

$R_w$ affects all $F_2$ nodes in a column-assembly and is driven by

$$dR_w/dt = -R_w + [\Sigma_{j \in F2} \, f(y_j)] \, f(v \, \| \, \underline{I}_w \, \| - \| \, \underline{P}_{1,w} \, \|) \; . \tag{5}$$

$v < 1$ is a vigilance parameter (Carpenter and Grossberg, 1987): for either (4) or (5) R > 0 at equilibrium just when the intersection between memory and input, $\underline{P}_1 = T \cap \underline{I}$, is relatively small, i.e. R > 0 when $v \, \| \, \underline{I} \, \| > \| \, \underline{P}_1 \, \|$. When the system is in operation, we fix $R_w = 0$ and input the pattern $\underline{I}_w = 0$. To recall the row in table C indexed by $\underline{u}$, we input $\underline{u}$ to all column-assemblies, and at equilibrium $x_{i,m} = \Sigma_{j \in F2} \, T_{ji} f(y_j)$. Thus $x_{i,m}$ represents the memory of the element in C corresponding to $\underline{u}$ and the column in C with the same label as the column-assembly. Winter (1988b) discusses recall dynamics in more detail.

At equilibrium in either training or operational mode only the winning $F_2$ node has $y_J \neq$ 0, so $\Sigma_j T_{ji} f(y_j) = T_{Ji}$ in (2). Hence $x_{i,k} = 0$ if $T_{Ji} = -I_{i,k}$, i.e. if memory and input mismatch; $|x_{i,k}| = 2$ if $T_{Ji} = I_{i,k}$, i.e. when memory and input match; and $|x_{i,k}| = 1$ if $T_{J,i} = 0$, $I_{i,k} \neq 0$ or if $T_{J,i} \neq 0$, $I_{i,k} = 0$. The $F_1$ output function h in (3) is defined so that $h(x) = 1$ if $x > 1$, $h(x) = -1$ if $x < -1$ and $h(x) = 0$ if $-1 \leq x \leq 1$. The output pattern $\underline{P}_1 = (h(x_1), ..., h(x_{n1}))$ reflects $\underline{T}_J \cap \underline{I}_k$, as $h(x_i) \neq 0$ only if $T_{Ji} = I_{i,k}$.

The adaptive filters $(B_{u,j})$ and $(B_{m,j})$ store normalized versions of those patterns on $F_{1,u}$ and $F_{1,m}$ which have stimulated the $j^{th}$ $F_2$ node. The evolution of $B_{ij}$ for $u \in F_{1,u}$ or $F_{1,m}$ is driven by

$$dB_{ji}/dt = f(y_j) \left[ h(x_i)/(1 + \Sigma_i h^2(x_i)) - B_{ij} \right]. \qquad (6)$$

On the other hand $(B_w)$ stores a normalized version of the experiment $\underline{w}$ which labels the entire column-assembly. Thus all nodes in a column-assembly share a common memory of $\underline{w}$,

$$dB_w/dt = f( \Sigma_{j \in F2} f(y_j) ) \left[ h(x_i)/(1 + \Sigma_i h^2(x_i)) - B_w \right], \qquad (7)$$

where $w \in F_{1,w}$.

The feedback filters $(T_{u,j})$, $(T_{m,j})$ and $(T_w)$ store exact memories of patterns on partitions of $F_1$:

$$dT_{ij}/dt = f(y_j) \left[ h(x_i) - T_{ij} \right] \qquad (8)$$

for $i \in F_{1,u}$, $F_{1,m}$, and

$$dT_w/dt = f( \Sigma_{j \in F2} f(y_j) ) \left[ h(x_i) - T_w \right] \qquad (9)$$

for $i \in F_{1,w}$. In operation long-term memory modification is suspended.

## EXPERIMENT

Here we report partial results from an experiment in which TIN-2 learns a state characterization table for an automaton that recognizes strings containing even numbers of

both 1's and 0's. More details can be found in Winter (1988b). For notational convenience in this section we will discuss patterns as if they were composed of 1's and 0's, but be aware that inside TIN-2 every 0 symbol is really a -1. Data is provided in the form of triples ($\underline{M}$, $\underline{U}$, $\underline{W}$) by a teacher; the data set for this example is given in Table 1. Data were presented to the net in the order shown. The net consisted of three column-assemblies. Each $F_2$ collection contained ten nodes. Although the strings that can be processed by an automaton of this type are in principle arbitrarily long, in practice some limitation on the length of training strings is necessary if for no other reason than that the memory capacity of a computer is finite. For this simple example Input and $F_1$ partitions contain eight nodes, but in order to have a special symbol to represent $\lambda$, strings are limited to at most six elements. With this restriction the $\lambda$ symbol can be distinguished from actual input strings through vigilance criteria. Other solutions to the problem of representing $\lambda$ are being investigated, but for now the special eight bit symbol, 00000011, is used to represent $\lambda$ in the strings $\lambda \cdot \underline{W}$.

The net was trained using fast-learning (Carpenter and Grossberg, 1987): a triple in Table 1 was presented to the net, and all nodes were allowed to come to their equilibrium values where they were held for about three long-term time units before the next triple was presented. Consider the processing that follows presentation of (0, 1, 0) the first datum in Table 1. The net can obtain equivalents to two C table entries from (0, 1, 0): the entry in row $\underline{u}$ = 10, column $\underline{w}$ = $\lambda$ and the entry in row $\underline{u}$ =1, column $\underline{w}$ = 0. The string 10 and the membership value 0 were displayed on the $\lambda$ assembly's input slabs, and in this case the $3^{rd}$ $F_2$ node learned the association among the two patterns. When the pattern (0, 1, 0) was input to other column-assemblies, one $F_2$ node (in this case the $9^{th}$ in column-assembly 1) learned to associate elements of the triple. Of course a side effect of this was that column-assembly 1 was labelled by $\underline{W}$ = 0 thereafter. When (1, 1, 1) was input next, node 9 in column-assembly 1 tried to respond to the new triple, all nodes in column-assembly 1 were then inhibited by a mismatch on $\underline{W}$, and finally node 1 on column-assembly 2 learned (1, 1, 1). From that point on column-assembly 2 was labelled by 1.

## LEARNING TRANSITIONS

The TIN-1 net (Winter, Ryan and Turner, 1987) is composed of i) a partitioned adaptive resonance circuit with dynamics similar to (2) - (9) for learning state transitions and ii) a Control Circuit which forces transitions once they have been learned. Transitions are unique in the sense that a previous state and current input completely determine the current state. The partitioned adaptive resonance circuit has three input fields: one for the previous state, one for the current input and one for the next state. TIN-1's $F_2$ nodes learn transitions by associating patterns in the three input fields. Once trained, TIN-1 processes strings sequentially, bit-by-bit.

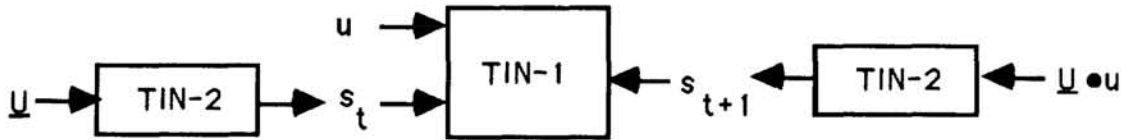

Figure 1.  Training TIN$^2$.

The architecture of TIN$^2$, the net that integrates TIN-2 and TIN-1, is shown in Figure 1. The system resorts to the TIN-2 nets only to learn transitions.  If TIN-2 has learned a C table in which examples of all transitions appear, TIN-1 can easily learn the automaton's state transitions.  A C table contains an example of a transition from state $s_i$ to state $s_j$ forced by current input u, if it contains i) a row labelled by a string $\underline{U}_i$ which leaves the automaton in $s_i$ after processing and ii) a row labelled by the string $\underline{U}_i \cdot u$ which leaves the automaton in $s_j$.  To teach TIN-1 the transition we simply present $\underline{U}_i$ to the lower TIN-2 in Figure 1, $\underline{U}_i \cdot u$ to the upper TIN-2 net and u to TIN-1.

## CONCLUSIONS

We have described a network, TIN-2, which learns the equivalent of state characterization tables (Gold, 1972).  The principle reasons for developing a neural net implementation are i) neural nets are intrinsically massively parallel and so provide a nice model for systems that must process large data sets, ii) although in the interests of brevity we have not stressed the point, neural nets are robust against noisy data, iii) neural nets like the partitioned adaptive resonance circuit have continuous time activity dynamics and so can be synchronized with other elements of a larger real-time system through simple scaling parameters, and iv) the continuous time dynamics and precise architectural specifications of neural nets provide a blueprint for hardware implementations.

We have also sketched a neural net, TIN$^2$, that learns state transitions by integrating TIN-2 nets with the TIN-1 net (Winter, Ryan and Turner, 1987).  When a complete state characterization table is available from TIN-2, TIN$^2$ can be taught transitions from examples of system behavior.  However, the ultimate goal of a net like this lies in developing a system that "operates acceptably" with a partial state characterization table. To operate acceptably TIN$^2$ must perform transitions correctly when it can, recognize when it cannot, signal for new data when it is required and expand the state charcterization table when it must.  Happily TIN$^2$ already provides the first two capabilities, and combinations of TIN$^2$ with rule-based controllers and with auxiliary control networks are currently being explored as approachws to satisfy the latter (Winter, 1988b).

Nets like TIN$^2$ may eventually prove useful as control elements in physical machines because sequential automata can respond to unpredictable environments with a wide range of behavior.  Even very simple automata can repeat activities and make decisions based upon environmental variations.  Currently, most physical machines that make decisions are dedicated to a single task; applying one to a new task requires re-programming by a

skilled technician. A programmer must, furthermore, determine a priori precisely which machine state - environment associations are significant enough to warrant insertion in the control structure of a given machine. $TIN^2$, on the other hand, is trained, not programmed, and can abstract significant associations from noisy input. It is a "blank slate" that learns the structure of a particular sequential machine from examples.

## References

D. Angluin, "Learning Regular Sets from Queries and Counterexamples", Information and Computation, 75 (2), 1987.

M. A. Arbib and E. G. Manes, "Machines in a Category: an Expository Introduction", SIAM Review, 16 (2), 1974.

M. A. Arbib and H. P. Zeiger, "On the Relevance of Abstract Algebra to Control Theory", Automatica, 5, 1969.

G. Carpenter and S. Grossberg, "A Massively Parallel Architecture for a Self-Organizing Neural Pattern Recognition Machine", Comput. Vision Graphics Image Process. 37 (54), 1987.

E. M. Gold, "System Identification Via State Characterization", Automatica, 8, 1972.

E. M. Gold, "Complexity of Automaton Identification from Given Data", Info. and Control, 37, 1978.

A. Neroda, "Linear Automaton Transformations", Proc. Am. Math. Soc., 9, 1958.

T. W. Ryan and C. L. Winter, "Variations on Adaptive Resonance", in Proc. 1st Intl. Conf. on Neural Networks, IEEE, 1987.

T. W. Ryan, C. L. Winter and C. J. Turner, "Dynamic Control of an Artificial Neural System: the Property Inheritance Network", Appl. Optics, 261 (23) 1987.

V. V. Tolat and B. Widrow, "An Adaptive Neural Net Controller with Visual Inputs", Neural Networks, 1, Suppl, 1988.

C. L. Winter, T. W. Ryan and C. J. Turner, "TIN: A Trainable Inference Network", in Proc. 1st Intl. Conf. on Neural Networks, 1987.

C. L. Winter, "An Adaptive Network that Flees Pursuit", Neural Networks, 1, Supp.1, 1988a.

C. L. Winter, "$TIN^2$: An Adaptive Controller", SAIC Tech. Rpt., SAIC, 5151 E. Broadway, Tucson, AZ, 85711, 1988b.

